# Learning via Gaussian Herding

**Koby Crammer**
Department of Electrical Enginering
The Technion
Haifa, 32000 Israel
koby@ee.technion.ac.il

**Daniel D. Lee**
Dept. of Electrical and Systems Engineering
University of Pennsylvania
Philadelphia, PA 19104
ddlee@seas.upenn.edu

## Abstract

We introduce a new family of online learning algorithms based upon constraining the velocity flow over a distribution of weight vectors. In particular, we show how to effectively *herd* a Gaussian weight vector distribution by trading off velocity constraints with a loss function. By uniformly bounding this loss function, we demonstrate how to solve the resulting optimization analytically. We compare the resulting algorithms on a variety of real world datasets, and demonstrate how these algorithms achieve state-of-the-art robust performance, especially with high label noise in the training data.

## 1   Introduction

Online learning algorithms are simple, fast, and require less memory compared to batch learning algorithms. Recent work has shown that they can also perform nearly as well as batch algorithms in many settings, making them quite attractive for a number of large scale learning problems [3]. The success of an online learning algorithm depends critically upon a tradeoff between fitting the current data example and regularizing the solution based upon some memory of prior hypotheses. In this work, we show how to incorporate regularization in an online learning algorithm by constraining the motion of weight vectors in the hypothesis space. In particular, we demonstrate how to use simple constraints on the velocity flow field of Gaussian-distributed weight vectors to regularize online learning algorithms. This process results in *herding* the motion of the Gaussian weight vectors to yield algorithms that are particularly robust to noisy input data.

Recent work has demonstrated how parametric information about the weight vector distribution can be used to guide online learning [1]. For example, confidence weighted (CW) learning maintains a Gaussian distribution over linear classifier hypotheses and uses it to control the direction and scale of parameter updates [9]. CW learning has formal guarantees in the mistake-bound model [7]; however, it can over-fit in certain situations due to its aggressive update rules based upon a separable data assumption. A newer online algorithm, Adaptive Regularization of Weights (AROW) relaxes this separable assumption, resulting in an adaptive regularization for each training example based upon its current confidence [8]. This regularization comes in the form of minimizing a bound on the Kullback-Leibler divergence between Gaussian distributed weight vectors.

Here we take a different microscopic view of the online learning process. Instead of reweighting and diffusing the weight vectors in hypothesis space, we model them as flowing under a velocity field given by each data observation. We show that for linear velocity fields, a Gaussian weight vector distribution will maintain its Gaussianity, with corresponding updates for its mean and covariance. The advantage of this approach is that we can incorporate different constraints and regularization on the resulting velocity fields to yield more robust online learning algorithms. In the remainder of this paper, we elucidate the details of our approach and compare its performance on a variety of experimental data.

These algorithms maintain a Gaussian distribution over possible weight vectors in hypothesis space. In traditional stochastic filtering, weight vectors are first reweighted according to how accurately they describe the current data observation. The remaining distribution is then subjected to random diffusion, resulting in a new distribution. When the reweighting factor depends linearly upon the weight vector in combination with a Gaussian diffusion model, a weight vector distribution will maintain its Gaussianity under such a transformation. The Kalman filter equations then yield the resulting change in the mean and covariance of the new distribution. Our approach, on the other hand, updates the weight vector distribution with each observation by herding the weight vectors using a velocity field. The differences between these two processes are shown in Fig. 1.

## 2 Background

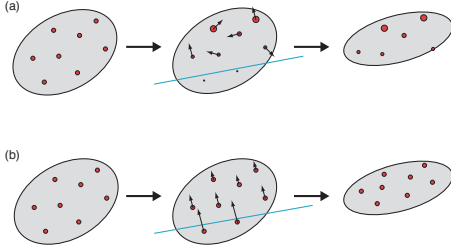

Figure 1: (a) Traditional stochastic filtering: weight vectors in the hypothesis space are reweighted according to the new observation and undergo diffusion resulting in a new weight vector distribution. (b) Herding via a velocity field: weights vectors flow in hypothesis space according to a constrained velocity field, resulting in a new weight vector distribution.

Consider the following online binary classification problem, that proceeds in rounds. On the $ith$ round the online algorithm receives an input $\boldsymbol{x}_i \in \mathbb{R}^d$ and applies its current prediction rule to make a prediction $\hat{y}_i \in \mathcal{Y}$, for the binary set $\mathcal{Y} = \{-1, +1\}$. It then receives the correct label $y_i \in \mathcal{Y}$ and suffers a loss $\ell(y_i, \hat{y}_i)$. At this point, the algorithm updates its prediction rule with the pair $(\boldsymbol{x}_i, y_i)$ and proceeds to the next round. A summary of online algorithms can be found in [2].

An initial description for possible online algorithms is provided by the family of passive-aggressive (PA) algorithms for linear classifiers [5]. The weight vector $\boldsymbol{w}_i$ at each round is updated with the current input $\boldsymbol{x}_i$ and label $y_i$, by optimizing:

$$\boldsymbol{w}_{i+1} = \arg\min_{\boldsymbol{w}} \frac{1}{2}\|\boldsymbol{w} - \boldsymbol{w}_i\|^2 + C\ell\left((\boldsymbol{x}_i, y_i), \boldsymbol{w}\right) , \tag{1}$$

where $\ell\left((\boldsymbol{x}_i, y_i), \boldsymbol{w}\right)$ is the squared- or hinge-loss function and $C > 0$ controls the tradeoff between optimizing the current loss and being close to the old weight vector. Eq. (1) can also be expressed in dual form, yielding the PA-II update equation:

$$\boldsymbol{w}_{i+1} = \boldsymbol{w}_i + \alpha_i y_i \boldsymbol{x}_i \quad , \quad \alpha_i = \left(\max\{0, 1 - y_i(\boldsymbol{w}_i^\top \boldsymbol{x}_i)\}\right) / \left(\|\boldsymbol{x}_i\|^2 + 1/C\right) . \tag{2}$$

The theoretical properties of this algorithm was analyzed by [5], and it was demonstrated on a variety of tasks (e.g. [3]).

Online confidence-weighted (CW) learning [9, 7], generalized the PA update principle to multivariate Gaussian distributions over the weight vectors $\mathcal{N}(\boldsymbol{\mu}, \Sigma)$ for binary classification. The mean $\boldsymbol{\mu} \in \mathbb{R}^d$ contains the current estimate for the best weight vector, whereas the Gaussian covariance matrix $\Sigma \in \mathbb{R}^{d \times d}$ captures the confidence in this estimate.

CW classifiers are trained according to a PA rule that is modified to track differences in Gaussian distributions. At each round, the new mean and covariance of the weight vector distribution is chosen by optimizing: $(\boldsymbol{\mu}_{i+1}, \Sigma_{i+1}) = \arg\min_{\boldsymbol{\mu}, \Sigma} D_{\text{KL}}\left(\mathcal{N}(\boldsymbol{\mu}, \Sigma) \| \mathcal{N}(\boldsymbol{\mu}_i, \Sigma_i)\right)$ such that $\Pr_{\boldsymbol{w} \sim \mathcal{N}(\boldsymbol{\mu}, \Sigma)}\left[y_i(\boldsymbol{w} \cdot \boldsymbol{x}_i) \geq 0\right] \geq \eta$.

This particular CW rule may over-fit since it guarantees a correct prediction with likelihood $\eta > 0.5$ at every round. A more recent alternative scheme called AROW (adaptive regularization of weight-vectors) [8] replaces the guaranteed prediction at each round with the following loss function: $(\boldsymbol{\mu}_{i+1}, \Sigma_{i+1}) = \arg\min_{\boldsymbol{\mu}, \Sigma} D_{\text{KL}}\left(\mathcal{N}(\boldsymbol{\mu}, \Sigma) \| \mathcal{N}(\boldsymbol{\mu}_i, \Sigma_i)\right) + \lambda_1 \ell_{\text{h}^2}(y_i, \boldsymbol{\mu} \cdot \boldsymbol{x}_i) + \lambda_2 \boldsymbol{x}_i^\top \Sigma \boldsymbol{x}_i$ ,where $\ell_{\text{h}^2}(y_i, \boldsymbol{\mu} \cdot \boldsymbol{x}_i) = \left(\max\{0, 1 - y_i(\boldsymbol{\mu} \cdot \boldsymbol{x}_i)\}\right)^2$ is the squared-hinge loss suffered using the weight vector $\boldsymbol{\mu}$ and $\lambda_1, \lambda_2 \geq 0$ are two tradeoff hyperparameters. AROW [8] has been shown to perform well in practice, especially for noisy data where CW severely overfits.

In this work, we take the view that the Gaussian distribution over weight vectors is modified by herding according to a velocity flow field. First we show that any change in a Gaussian distributed random variable can be related to a linear velocity field:

**Theorem 1** *Assume that the random variable (r.v.)* $\mathbf{W}$ *is distributed according to a Gaussian distribution,* $\mathbf{W} \sim \mathcal{N}\left(\boldsymbol{\mu}, \Sigma\right)$ ,

1. *The r.v.* $\mathbf{U} = A\mathbf{W} + \boldsymbol{b}$ *also has a Gaussian distribution,* $\mathbf{U} \sim \mathcal{N}\left(\boldsymbol{b} + A\boldsymbol{\mu}, A\Sigma A^{\top}\right)$ .
2. *Assume that a r.v.* $\mathbf{U}$ *is distributed according to a Gaussian distribution,* $\mathbf{U} \sim \mathcal{N}\left(\tilde{\boldsymbol{\mu}}, \tilde{\Sigma}\right)$ . *Then there exists $A$ and $\boldsymbol{b}$ such that the following linear relation holds,* $\mathbf{U} = A\mathbf{W} + \boldsymbol{b}$ .
3. *Let $\Upsilon$ be any orthogonal matrix $\Upsilon^{\top} = \Upsilon^{-1}$ and define* $\mathbf{U} = \Sigma^{\frac{1}{2}} \Upsilon \Sigma^{-\frac{1}{2}} \left(\mathbf{W} - \boldsymbol{\mu}\right) + \boldsymbol{\mu}$, *then both $\mathbf{U}$ and $\mathbf{W}$ have the same distribution.*

**Proof:** The first property follows easily from linear systems theory. The second property is easily shown by taking: $A = \tilde{\Sigma}^{\frac{1}{2}} \Sigma^{-\frac{1}{2}}$ and $\boldsymbol{b} = \tilde{\boldsymbol{\mu}} - \tilde{\Sigma}^{\frac{1}{2}} \Sigma^{-\frac{1}{2}} \boldsymbol{\mu}$ . Similarly, for the third property, it suffices to show that $\mathrm{E}\left[\mathbf{U}\right] = \Sigma^{\frac{1}{2}} \Upsilon \Sigma^{-\frac{1}{2}} \left(\mathrm{E}\left[\mathbf{W}\right] - \boldsymbol{\mu}\right) + \boldsymbol{\mu} = \boldsymbol{\mu}$ , and $\mathrm{Cov}\left(\mathbf{U}\right) = \mathrm{E}\left[\left(\mathbf{U} - \boldsymbol{\mu}\right)\left(\mathbf{U} - \boldsymbol{\mu}\right)^{\top}\right] = \Sigma^{\frac{1}{2}} \Upsilon \Sigma^{-\frac{1}{2}} \mathrm{E}\left[\left(\mathbf{W} - \boldsymbol{\mu}\right)\left(\mathbf{W} - \boldsymbol{\mu}\right)^{\top}\right] \Sigma^{-\frac{1}{2}} \Upsilon^{\top} \Sigma^{\frac{1}{2}} = \Sigma^{\frac{1}{2}} \Upsilon \Sigma^{-\frac{1}{2}} \Sigma \Sigma^{-\frac{1}{2}} \Upsilon^{\top} \Sigma^{\frac{1}{2}} = \Sigma^{\frac{1}{2}} \Upsilon \Upsilon^{\top} \Sigma^{\frac{1}{2}} = \Sigma^{\frac{1}{2}} \Sigma^{\frac{1}{2}} = \Sigma$ . ∎

Thus, the transformation $\mathbf{U} = A\mathbf{W} + \boldsymbol{b}$ can be viewed as a velocity flow resulting in a change of the underlying Gaussian distribution of weight vectors. On the other hand, this microscopic view of the underlying velocity field contains *more* information than merely tracking the mean and covariance of the Gaussian. This can be seen since many different velocity fields result in the same overall mean and covariance. In the next section, we show how we can define new online learning algorithms by considering various constraints on the overall velocity field. These new algorithms optimize a loss function by constraining the parameters of this velocity field.

## 3   Algorithms

Our algorithms maintain a distribution, or infinite collection of weight vectors $\{\mathbf{W}_i\}$ for each round $i$. Given an instance $\boldsymbol{x}_i$ it outputs a prediction based upon the majority of these weight vectors. Each weight vector $\mathbf{W}_i$ is then individually updated to $\mathbf{W}_{i+1}$ according to a generalized PA rule,

$$\mathbf{W}_{i+1} = \arg\min_{\mathbf{W}} \mathcal{C}_i\left(\mathbf{W}\right) \quad \text{where} \quad \mathcal{C}_i\left(\mathbf{W}\right) = \frac{1}{2}\left(\mathbf{W} - \mathbf{W}_i\right)^{\top} \Sigma_i^{-1}\left(\mathbf{W} - \mathbf{W}_i\right) + C\ell\left(\left(\boldsymbol{x}_i, y_i\right), \mathbf{W}\right) , \quad (3)$$

and $\Sigma_i$ is a PSD matrix that will be defined shortly. In fact, we assume that $\Sigma_i$ is invertible and thus PD.

Clearly, it is impossible to maintain and update an infinite set of vectors, and thus we employ a parametric density $f_i(\mathbf{W}_i; \theta_i)$ to weight each vector. In general, updating each individual weight-vector using some rule (such as the PA update) will modify the parametric family. We thus employ a Gaussian parametric density with $\mathbf{W} \sim \mathcal{N}\left(\boldsymbol{\mu}_i, \Sigma_i\right)$, and update the distribution collectively,

$$\mathbf{W}_{i+1} = A_i \mathbf{W}_i + \boldsymbol{b}_i ,$$

where $A_i \in \mathbb{R}^{d \times d}$ represents stretching and rotating the distribution, and the $\boldsymbol{b}_i \in \mathbb{R}^d$ is an overall translation. Incorporating this linear transformation, we minimize the average of Eq. (3) with respect to the current distribution,

$$\left(A_i, \boldsymbol{b}_i\right) = \arg\min_{A, \boldsymbol{b}} \mathrm{E}_{\mathbf{W}_i \sim \mathcal{N}(\boldsymbol{\mu}_i, \Sigma_i)}\left[\mathcal{C}_i\left(A\mathbf{W}_i + \boldsymbol{b}\right)\right] . \quad (4)$$

We derive the algorithm by computing the expectation Eq. (4) starting with the first regularization term of Eq. (3). After some algebraic manipulations and using the first property of Theorem 1 to write $\boldsymbol{\mu} = A\boldsymbol{\mu}_i + \boldsymbol{b}_i$ we get the expected value for the first term of Eq. (3) in terms of $\boldsymbol{\mu}$ and $A$,

$$\frac{1}{2}\left(\boldsymbol{\mu} - \boldsymbol{\mu}_i\right)^{\top} \Sigma_i^{-1}\left(\boldsymbol{\mu} - \boldsymbol{\mu}_i\right) + \frac{1}{2}\mathrm{Tr}\left(\left(A - I\right)^{\top} \Sigma_i^{-1}(A - I)\Sigma_i\right) . \quad (5)$$

Next, we focus on the expectation of the loss function in their second term of Eq. (3).

### 3.1   Expectation of the Loss Function

We consider the expectation,

$$\mathrm{E}_{\mathbf{W}_i \sim \mathcal{N}(\boldsymbol{\mu}_i, \Sigma_i)}\left[\ell\left(\left(\boldsymbol{x}_i, y_i\right), A\mathbf{W}_i + \boldsymbol{b}\right)\right] \quad (6)$$

In general, there is no closed form solution for this expectation, and instead we seek for an appropriate approximation or bound. For simplicity we consider binary classification, denote the signed margin by $M = y_i(\mathbf{W}^\top \boldsymbol{x})$ and write $\ell\left((\boldsymbol{x}, y), \mathbf{W}\right) = \ell(M)$.

If the loss is relatively concentrated about its mean, then the loss of the expected weight-vector $\boldsymbol{\mu}$ is a good proxy for Eq. (6). Formally, we can define

**Definition 1** *Let $\mathcal{F} = \{f(M; \theta) \ : \ \theta \in \Theta\}$ be a family of density functions. A loss function is uniformly $\lambda$-bounded in expectation with respect to $\mathcal{F}$ if there exists $\lambda > 0$ such that for all $\theta \in \Theta$ we have that, $\mathrm{E}\left[\ell\left(M\right)\right] \leq \ell\left(\mathrm{E}\left[M\right]\right) + \frac{\lambda}{2}\mathrm{E}\left[\left(M - \mathrm{E}\left[M\right]\right)^2\right]$, where all expectations are with respect $M \sim f(M; \theta)$.*

We note in passing that if the loss function $\ell$ is convex with respect to $\mathbf{W}$ we always have that, $\mathrm{E}\left[\ell\left(M\right)\right] \geq \ell\left(\mathrm{E}\left[M\right]\right)$. For Gaussian distributions we have that $\Theta = \{\boldsymbol{\mu}, \Sigma\}$ and a loss function $\ell$ is uniformly $\lambda$-bounded in expectation if there exists a $\lambda$ such that, $\mathrm{E}_{\mathcal{N}(\boldsymbol{\mu}, \Sigma)}\left[\ell\left((\boldsymbol{x}, y), \mathbf{W}\right)\right] \leq \ell\left((\boldsymbol{x}, y), \mathrm{E}\left[\mathbf{W}\right]\right) + \frac{\lambda}{2}\boldsymbol{x}^\top \Sigma \boldsymbol{x}$. We now enumerate some particular cases where losses are uniformly $\lambda$-bounded.

**Proposition 2** *Assume that the loss function $\ell(M)$ has a bounded second derivative, $\ell''(M) \leq \lambda$ then $\ell$ is uniformly $\lambda$-bounded in expectation.*

**Proof:** Applying the Taylor expansion about $M = \mathrm{E}\left[M\right]$ we get, $\ell\left(M\right) = \ell\left(\mathrm{E}\left[M\right]\right) + \left(M - \mathrm{E}\left[M\right]\right)\ell'\left(\mathrm{E}\left[M\right]\right) + \frac{1}{2}\left(M - \mathrm{E}\left[M\right]\right)^2 \ell''\left(\xi\right)$, for some $\xi \in [M, \mathrm{E}\left[M\right]]$. Taking the expectation of both sides and bounding $\ell''(\xi) \leq \lambda$ concludes the proof. ∎

For example, the squared loss $\frac{1}{2}\left(y - M^\top \boldsymbol{x}\right)^2$ is uniformly ($\lambda =$)1-bounded in expectation since its second derivative is bounded by unity (1). Another example is the log-loss, $\log(1 + \exp(-M))$, being uniformly $1/4$-bounded in expectation. Note that the popular hinge and squared-hinge loss are not even differentiable at $M = 1$. Nevertheless, we can show explicitly that indeed both are uniformly $\lambda$-bounded, though the proof is omitted here due to space considerations. To conclude, for uniformly $\lambda$-bounded loss functions, we bound Eq. (6) with $\ell\left((\boldsymbol{x}_i, y_i), \boldsymbol{\mu}\right) + \frac{\lambda}{2}\boldsymbol{x}_i^\top A\Sigma_i A^\top \boldsymbol{x}_i$. Thus, our online algorithm minimizes the following bound on Eq. (4), with a change of variables from the pair $(A, \boldsymbol{b})$ to the pair $(A, \boldsymbol{\mu})$, where $\boldsymbol{\mu}$ is the mean of the *new* distribution,

$$(A_i, \boldsymbol{\mu}_{i+1}) = \arg\min_{A, \boldsymbol{\mu}} \qquad \frac{1}{2}\left(\boldsymbol{\mu} - \boldsymbol{\mu}_i\right)^\top \Sigma_i^{-1}\left(\boldsymbol{\mu} - \boldsymbol{\mu}_i\right) + C\ell\left((\boldsymbol{x}_i, y_i), \boldsymbol{\mu}\right) + \qquad (7)$$

$$\frac{1}{2}\mathrm{Tr}\left((A - I)^\top \Sigma_i^{-1}(A - I)\Sigma_i\right) + \frac{C\lambda}{2}\boldsymbol{x}_i^\top A\Sigma_i A^\top \boldsymbol{x}_i \qquad (8)$$

In the next section we derive an analytic solution for the last problem. We note that, similar to AROW, it is decomposed into two additive terms: Eq. (7) which depends only on $\boldsymbol{\mu}$ and Eq. (8) which depends only on $A$.

## 4 Solving the Optimization Problem

We consider here the squared-hinge loss, $\ell\left((\boldsymbol{x}, y), \boldsymbol{\mu}\right) = \left(\max\{0, 1 - y(\boldsymbol{\mu}^\top \boldsymbol{x})\}\right)^2$, reducing Eq. (7) to a generalization of PA-II in Mahalanobis distances (see Eq. (2)),

$$\boldsymbol{\mu}_{i+1} = \boldsymbol{\mu}_i + \alpha_i y_i \boldsymbol{x}_i \ , \ \alpha_i = \left(\max\{0, 1 - y_i(\boldsymbol{\mu}_i^\top \boldsymbol{x}_i)\}\right) / \left(\boldsymbol{x}_i^\top \Sigma_i \boldsymbol{x}_i + 1/C\right) \ , \qquad (9)$$

We now focus on minimizing the second term (Eq. (8)) which depends solely on $A_i$. For simplicity we assume $\lambda = 1$ and consider two cases.

### 4.1 Diagonal Covariance Matrix

We first assume that both $\Sigma_i$ and $A$ are diagonal, and thus also $\Sigma_{i+1}$ is diagonal, and thus $\Sigma_i, \Sigma_{i+1}$ and $A$ commute with each other. Eq. (8) then becomes, $\frac{1}{2}\mathrm{Tr}\left((A - I)^\top (A - I)\right) + \frac{C}{2}\boldsymbol{x}_i^\top A\Sigma_i A^\top \boldsymbol{x}_i$. Denote the r*th* diagonal element of $\Sigma_i$ by $(\Sigma_i)_{r,r}$ and the r*th* diagonal element of $A$ by $(A)_{r,r}$. The

last equation becomes, $\sum_r \frac{1}{2}((A)_{r,r} - 1)^2 + \frac{C}{2} \sum_r x_{i,r}^2 (A)_{r,r}^2 (\Sigma_i)_{r,r}$ Taking the derivative with respect to $(A)_{r,r}$ we get,

$$(A_i)_{r,r} = 1 / \left(1 + Cx_{i,r}^2 (\Sigma_i)_{r,r}\right) \quad \Rightarrow \quad (\Sigma_{i+1})_{r,r} = (\Sigma_i)_{r,r} / \left(1 + Cx_{i,r}^2 (\Sigma_i)_{r,r}\right)^2 . \quad (10)$$

The last equation is well-defined since the denominator is always greater than or equal to 1.

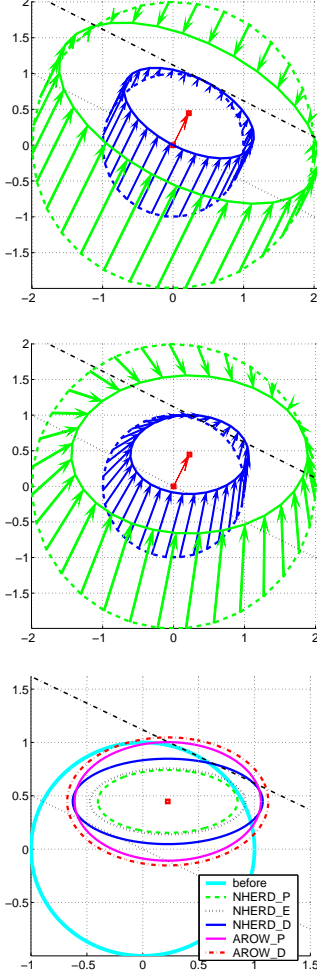

### 4.2 Full Covariance Matrix

Expanding Eq. (8) we get $\frac{1}{2}\left(\text{Tr}\left(A^\top \Sigma_i^{-1} A\Sigma_i\right) - \text{Tr}\left(\Sigma_i^{-1} A\Sigma_i\right)\right.$ $\left.+\text{Tr}\left(\Sigma_i^{-1}\Sigma_i\right) - \text{Tr}\left(A^\top \Sigma_i^{-1}\Sigma_i\right)\right) + \frac{C}{2}\boldsymbol{x}_i^\top A\Sigma_i A^\top \boldsymbol{x}_i$. Setting the derivative of the last equation with respect to $A$ we get, $\Sigma_i^{-1} A\Sigma_i - I + C\boldsymbol{x}_i\boldsymbol{x}_i^\top A\Sigma_i = 0$ . We multiply both terms by $\Sigma_i^{-1}$ (right) and combine terms, $\left(\Sigma_i^{-1} + C\boldsymbol{x}_i\boldsymbol{x}_i^\top\right) A = \Sigma_i^{-1}$ , Yielding,

$$A_i = \left(\Sigma_i^{-1} + C\boldsymbol{x}_i\boldsymbol{x}_i^\top\right)^{-1}\Sigma_i^{-1} . \quad (11)$$

To get $\Sigma_{i+1}$ we first compute its inverse, $\Sigma_{i+1}^{-1} = \left(A\Sigma_i A^\top\right)^{-1}$ . Substituting Eq. (11) in the last equation we get,

$$\Sigma_{i+1}^{-1} = \left(A\Sigma_i A^\top\right)^{-1} = \Sigma_i^{-1} + \left(2C + C^2 \boldsymbol{x}_i^\top \Sigma_i \boldsymbol{x}_i\right)\boldsymbol{x}_i\boldsymbol{x}_i^\top \quad (12)$$

Finally, using the Woodbury identity [12] to compute to updated covariance matrix,

$$\Sigma_{i+1} = \Sigma_i - \Sigma_i \boldsymbol{x}_i\boldsymbol{x}_i^\top \Sigma_i \left(C^2\boldsymbol{x}_i\Sigma_i\boldsymbol{x}_i^\top + 2C\right) / \left((1 + C\boldsymbol{x}_i^\top \Sigma_i \boldsymbol{x}_i)^2\right) . \quad (13)$$

We call the above algorithms NHERD for Normal (Gaussian) Herd. A pseudocode of the algorithm appears in Alg. 3.

### 4.3 Discussion

Both our update of $\Sigma_{i+1}$ in Eq. (12) and the update of AROW (see eq. (8) of [8] ) have the same structure of adding $\gamma_i \boldsymbol{x}_i\boldsymbol{x}_i^\top$ to $\Sigma_i$. AROW sets $\gamma_i = C$ while our update sets $\gamma_i = 2C + C^2\boldsymbol{x}_i\Sigma_i\boldsymbol{x}_i^\top$. In this aspect, the NHERD update is more aggressive as it increases the eigenvalues of $\Sigma_i^{-1}$ at a faster rate. Furthermore, its update rate is not constant and depends linearly on the current variance of the margin $\boldsymbol{x}_i^\top \Sigma_i \boldsymbol{x}_i$; the higher the variance, the faster the eigenvalues of $\Sigma_i$ decrease. Lastly, we note that the update matrix $A_i$ can be written as a product of two terms, one depends on the covariance matrix before the update and the other on the covariance matrix after an AROW update. Formally, let $\tilde{\Sigma}_{i+1}$ be the covariance matrix after updated using the *AROW* rule, that is, $\tilde{\Sigma}_{i+1} = \left(\Sigma_i^{-1} + C\boldsymbol{x}_i\boldsymbol{x}_i^\top\right)$ (see eq. (8) of [8] ). From Eq. (11) we observe that $A_i = \tilde{\Sigma}_{i+1}^{-1}\Sigma_i$, which means that NHERD modifies $\Sigma_i$ if and only if AROW modifies $\Sigma_i$.

The diagonal updates of AROW and NHERD share similar properties. [8] did not specify the specific update for this case, yet using a similar derivation of Sec. 4.1 we get that the AROW update for diagonal matrices $\tilde{\Sigma}_{i+1}$ is $\left(\tilde{\Sigma}_{i+1}\right)_{r,r} = (\Sigma_i)_{r,r} / \left(1 + Cx_{i,r}^2 (\Sigma_i)_{r,r}\right)$ . Taking the ratio between the r$th$

Figure 2: Top and center panels: an illustration of the algorithm's update (see text). Bottom panel: an illustration of a single update for the five algorithms. The cyan ellipse represents the weight vector distribution before the example is observed. The red-square represents the mean of the updated distribution and the five ellipses represents the covariance of each of the algorithm after given the data example $((1, 2), +1)$. The ordering of the area of the five ellipses correlates well with the performance of the algorithms.

element of Eq. (10) and the last equation we get, $\left(\tilde{\Sigma}_{i+1}\right)_{r,r} / (\Sigma_{i+1})_{r,r} = 1 + Cx_{i,r}^2 (\Sigma_i)_{r,r} \geq 1$ .

To conclude, the update of NHERD for diagonal covariance matrices is also more aggressive than AROW as it increases the (diagonal) elements of its inverse faster than AROW.

An illustration of the two updates appears in Fig. 2 for a problem in a planar 2-dimensional space. The Gaussian distribution before the update is isotropic with mean $\boldsymbol{\mu} = (0,0)$ and $\Sigma = I_2$. Given the input example $\boldsymbol{x} = (1,2), y = 1$ we computed both $A$ and $\boldsymbol{b}$ for both the full (top panel) and diagonal (center panel) update. The plot illustrates the update of the mean vector (red square), weight vectors with unit norm $\|\boldsymbol{w}\| = 1$ (blue), and weight vectors with norm of 2, $\|\boldsymbol{w}\| = 2$ (green).

**Parameter:** $C > 0$
**Initialize:** $\boldsymbol{\mu}_1 = \mathbf{0}$ , $\Sigma_1 = I$
**for** $i = 1, \ldots, m$ **do**
  Get input example $\boldsymbol{x}_i \in \mathbb{R}^d$
  Predict $\hat{y}_i = \text{sign}(\boldsymbol{\mu}_i^\top \boldsymbol{x}_i)$
  Get true label $y_i$ and suffer loss 1 if $\hat{y}_i \neq y_i$
  **if** $y_i(\boldsymbol{\mu}_i^\top \boldsymbol{x}_i) \leq 1$ **then**
    Set $\boldsymbol{\mu}_{i+1} = \boldsymbol{\mu}_i + y_i \frac{\max\{0, 1 - y_i(\boldsymbol{\mu}_i^\top \boldsymbol{x}_i)\}}{\boldsymbol{x}_i^\top \Sigma_i \boldsymbol{x}_i + \frac{1}{C}} \Sigma_i \boldsymbol{x}_i$  (Eq. (9))
    **Full Covariance:**
      Set $\Sigma_{i+1} = \Sigma_i - \Sigma_i \boldsymbol{x}_i \boldsymbol{x}_i^\top \Sigma_i \frac{C^2 \boldsymbol{x}_i \Sigma_i \boldsymbol{x}_i^\top + 2C}{(1 + C \boldsymbol{x}_i \Sigma_i \boldsymbol{x}_i^\top)^2}$  (Eq. (13))
    **Diagonal Covariance:**
      Set $(\Sigma_{i+1})_{r,r}$ for $r = 1 \ldots d$ using Eq. (14)
  **end if**
**end for**
**Return:** $\boldsymbol{\mu}_{m+1}$ , $\Sigma_{m+1}$

Figure 3: Normal Herd (NHERD)

The ellipses with dashed lines illustrate the weights before the update, and ellipses with solid lines illustrate the weight-vectors after the update. All the weight vectors above the black dotted line classify the example correctly and the ones above the dashed lines classify the example with margin of at least unit 1. The arrows connecting weight-vectors from the dashed ellipses to solid ellipses illustrate the update of individual weight-vectors with the linear transformation $\boldsymbol{w} \leftarrow A_i(\boldsymbol{w} - \boldsymbol{\mu}_i) + \boldsymbol{\mu}_{i+1}$.

In both updates the current mean $\boldsymbol{\mu}_i$ is mapped to the next mean $\boldsymbol{\mu}_{i+1}$. The full update "shrinks" the covariance in the direction orthogonal to the example $y_i \boldsymbol{x}_i$; vectors close to the margin of unit 1 are modified less than vectors far from this margin; vectors with smaller margin are updated more aggressively then vectors with higher margin; even vectors that classify the example correctly with large margin of at least one are updated, such that their margin is shrunk. This is a consequence of the linear transformation that ties the update between all weight-vectors. The diagonal update, as designed, maintains a diagonal matrix, yet shrinks the matrix more in the directions that are more "orthogonal" to the example.

We note in passing that for all previous CW algorithms [7] and AROW [8], a closed form solution for diagonal matrices was not provided. Instead these papers proposed to diagonalize either $\Sigma_{i+1}$ (called `drop`) or $\Sigma_{i+1}^{-1}$ (called `project`) which was then inverted. Together with the exact solution of Eq. (10) we get the following three alternative solutions for diagonal matrices,

$$(\Sigma_{i+1})_{r,r} = \begin{cases} (\Sigma_i)_{r,r} / \left( \left(1 + Cx_{i,r}^2 (\Sigma_i)_{r,r}\right)^2 \right) & \texttt{exact} \\ 1 / \left( (1/(\Sigma_i)_{r,r}) + \left(2C + C^2 \boldsymbol{x}_i^\top \Sigma_i \boldsymbol{x}_i^\top\right) x_{i,r}^2 \right) & \texttt{project} \\ (\Sigma_i)_{r,r} - \left((\Sigma_i)_{r,r} x_{i,r}\right)^2 \frac{\left(C^2 \boldsymbol{x}_i \Sigma_i \boldsymbol{x}_i^\top + 2C\right)}{(1 + C \boldsymbol{x}_i \Sigma_i \boldsymbol{x}_i^\top)^2} & \texttt{drop} \end{cases} \quad (14)$$

We investigate these formulations in the next section. Finally, we note that similarly to CW and AROW, algorithms that employ full matrices can be incorporated with Mercer kernels [11, 14], while to the best of our knowledge, the diagonal versions can not.

## 5 Empirical Evaluation

We evaluate NHERD on several popular datasets for document classification, optical character recognition (OCR), phoneme recognition, as well as on action recognition in video. We compare our new algorithm NHERD with the AROW [8] algorithm, which was found to outperform other baselines [8]: the perceptron algorithm [13], Passive-Aggressive (PA) [5], confidence weighted learning (CW) [9, 7] and second order perceptron [1] on these datasets. For both NHERD and AROW we used the three diagonalization schemes, as mentioned in Eq. (14) in Sec. 4.3. Since AROW `Project` and AROW `Exact` are equivalent we omit the latter, yielding a total of five algorithms: NHERD_$\{P, D, E\}$ for `Project, Drop, Exact` and similarly AROW_$\{P, D\}$.

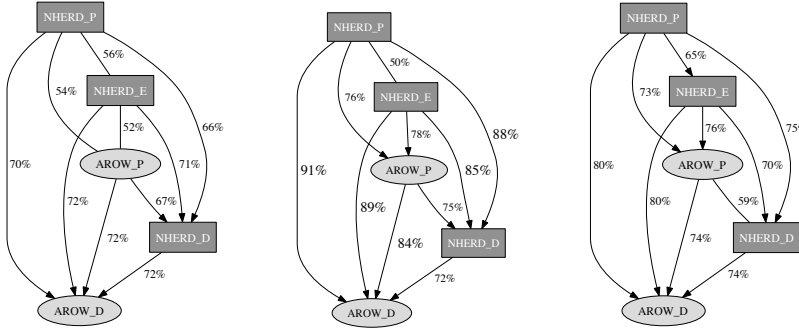

Figure 4: Performance comparison between algorithms. Each algorithm is represented by a vertex. The weight of an edge between two algorithms is the fraction of datasets in which the top algorithm achieves lower test error than the bottom algorithm. An edge with no head indicates a fraction lower than 60% and a bold edge indicates a fraction greater than 80%. Graphs (left to right) are for noise levels of 0%, 10%, and 30%.

Although NHERD and AROW are designed primarily for binary classification, we can modify them for use on multi-class problems as follows. Following [4], we generalize binary classification and assume a feature function $\boldsymbol{f}(\boldsymbol{x}, y) \in \mathbb{R}^d$ mapping instances $\boldsymbol{x} \in \mathcal{X}$ and labels $y \in \mathcal{Y}$ into a common space. Given a new example, the algorithm predicts $\hat{y} = \arg\max_z \boldsymbol{\mu} \cdot \boldsymbol{f}(\boldsymbol{x}, z)$, and suffers a loss if $y \neq \hat{y}$. It then computes the difference vector $\boldsymbol{\Delta} = \boldsymbol{f}(\boldsymbol{x}, y) - \boldsymbol{f}(\boldsymbol{x}, y')$ for $y' = \arg\max_{z \neq y} \boldsymbol{f}(\boldsymbol{x}, y')$ which replaces $y\boldsymbol{x}$ in NHERD (Alg. 3).

We conducted an empirical study using the following datasets. First are datasets from [8]: 36 binary document classification data, and 100 binary OCR data (45 all-pairs of both USPS and MNIST and 1-vs-rest of MNIST). Secondly, we used the nine multi-category document classification datasets used by [6]. Third, we conducted experiments on a TIMIT phoneme classification task. Here we used an experimental setup similar to [10] and mapped the 61 phonetic labels into 48 classes. We then picked 10 pairs of classes to construct binary classification tasks. We focused mainly on unvoiced phonemes where there is no underlying harmonic source and whose instantiations are noisy. The ten binary classification problems are identified by a pair of phoneme symbols (one or two Roman letters). For each of the ten pairs we picked $1,000$ random examples from both classes for training and $4,000$ random examples for a test set. These signals were then preprocessed by computing mel-frequency cepstral coefficients (MFCCs) together with first and second derivatives and second order interactions, yielding a feature vector of 902 dimensions. Lastly, we also evaluated our algorithm on an action recognition problem in video under four different conditions. There are about 100 samples for each of 6 actions. Each sample is represented using a set of 575 positive real localized spectral content filters from the videos. This yields a total of 156 datasets.

Each result for the text datasets was averaged over 10-fold cross-validation, otherwise a fixed split into training and test sets was used. Hyperparameters ($C$ for NHERD and $r$ for ARROW) and the number of online iterations (up to 20) were optimized using a single randomized run. In order to observe each algorithm's ability to handle non-separable data, we performed each experiment using various levels of artificial label noise, generated by independently flipping binary labels.

**Results:** We first summarize the results on all datasets excluding the video recognition dataset in Fig. 4, where we computed the number of datasets for which one algorithm achieved a lower test error than another algorithm. The results of this tournament between algorithms is presented as a winning percentage. An edge between two algorithms shows the fraction of the 155 datasets for which the algorithm on top had lower test error than the other algorithm. The three panels correspond to three varying noise levels, from 0%,10% and 30%.

We observe from the figure that Project generally outperforms Exact which in turn outperforms Drop. Furthermore, NHERD outperforms AROW, in particular NHERD_P outperforms AROW_P and NHERD_D outperforms AROW_D. These relations become more prominent when labeling noise is increased in the training data. The right panel of Fig. 2 illustrates a single update of each of the five algorithms: AROW_D, AROW_D, NHERD_D, NHERD_E, NHERD_P. Each of the five ellipses represents the Gaussian weight vector distribution after a single update on an example

by each of the five algorithms. Interestingly, the resulting volume (area) of different ellipses roughly correspond to the overall performance of the algorithms. The best update – NHERD_P – has the smallest ellipse (with lowest-entropy), and the update with the worst performance – AROW_D – has the largest, highest-entropy ellipse.

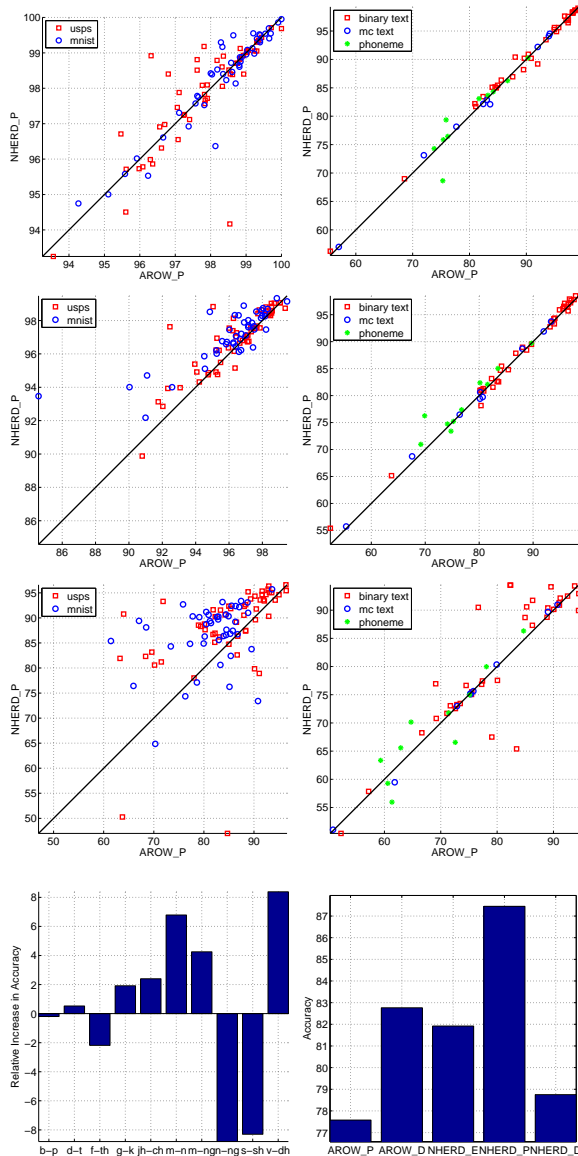

Figure 5: Three top rows: Accuracy on OCR (left) and text and phoneme (right) classification. Plots compare performance between NHERD_P and AROW_P. Markers above the line indicate superior NHERD_P performance and below the line superior AROW_P performance. Label noise increases from top to bottom: 0%, 10% and 30%. NHERD_P improves relative to AROW_P as noise increases. Bottom left: relative accuracy improvment of NHERD_P over AROW_P on the ten phoneme classification tasks. Bottom right: accuracy of five algorithms on the video data. In both cases NHERD_P is superior

More detailed results for NHERD_P and AROW_P, the overall best performing algorithms, are compared in Fig. 5. NHERD_P and AROW_P are comparable when there is no added noise, with NHERD_P winning a majority of the time. As label noise increases (moving top-to-bottom in Fig. 5) NHERD_P holds up remarkably well. In almost every high noise evaluation, NHERD_P improves over AROW_P (*as well as all other baselines, not shown*). The bottom-left panel of Fig. 5 shows the relative improvment in accuracy of NHERD_P over AROW_P on the ten phoneme recognition tasks with additional 30% label noise. The ten tasks are ordered according to their statistical significance according to McNemar's test. The results for the seven right tasks are statistically significant with a p-value less then 0.001. NHERD_P outperforms AROW_P five times and underperforms twice on these seven significant tests. Finally, the bottom-right panel shows the 10-fold accuracy of the five algorithms over the video data, where clearly NHERD_P outperforms all other algorithms by a wide margin.

**Conclusions:** We have seen how to incorporate velocity constraints in an online learning algorithm. In addition to tracking the mean and covariance of a Gaussian weight vector distribution, regularization of the linear velocity terms are used to *herd* the normal distribution in the learning process. By bounding the loss function with a quadratic term, the resulting optimization can be solved analytically, resulting in the NHERD algorithm. We empirically evaluated the performance of NHERD on a variety of experimental datasets, and show that the projected NHERD algorithm generally outperforms all other online learning algorithms on these datasets. In particular, NHERD is very robust when random labeling noise is present during training.

**Acknowledgments:** KC is a Horev Fellow, supported by the Taub Foundations. This work was also supported by German-Israeli Foundation grant GIF-2209-1912.

# References

[1] Nicoló Cesa-Bianchi, Alex Conconi, and Claudio Gentile. A second-order perceptron algorithm. *Siam Journal of Commutation*, 34(3):640–668, 2005.

[2] Nicolo Cesa-Bianchi and Gabor Lugosi. *Prediction, Learning, and Games*. Cambridge University Press, New York, NY, USA, 2006.

[3] G. Chechik, V. Sharma, U. Shalit, and S. Bengio. An online algorithm for large scale image similarity learning. In *NIPS*, 2009.

[4] Michael Collins. Discriminative training methods for hidden markov models: Theory and experiments with perceptron algorithms. In *EMNLP*, 2002.

[5] K. Crammer, O. Dekel, J. Keshet, S. Shalev-Shwartz, and Y. Singer. Online passive-aggressive algorithms. *JMLR*, 7:551–585, 2006.

[6] K. Crammer, M. Dredze, and A. Kulesza. Multi-class confidence weighted algorithms. In *EMNLP*, 2009.

[7] K. Crammer, M. Dredze, and F. Pereira. Exact confidence-weighted learning. In *NIPS 22*, 2008.

[8] K. Crammer, A. Kulesza, and M. Dredze. Adaptive regularization of weighted vectors. In *Advances in Neural Information Processing Systems 23*, 2009.

[9] M. Dredze, K. Crammer, and F. Pereira. Confidence-weighted linear classification. In *ICML*, 2008.

[10] A. Gunawardana, M. Mahajan, A Acero, and Pl att J. C. Hidden conditional random fields for phone classification. In *Proceedings of ICSCT*, 2005.

[11] J. Mercer. Functions of positive and negative type and their connection with the theory of integral equations. *Philos. Trans. Roy. Soc. London A*, 209:415–446, 1909.

[12] K. B. Petersen and M. S. Pedersen. The matrix cookbook, 2007.

[13] F. Rosenblatt. The perceptron: A probabilistic model for information storage and organization in the brain. *Psychological Review*, 65:386–407, 1958.

[14] Bernhard Schölkopf and Alexander J. Smola. *Learning with Kernels: Support Vector Machines, Regularization, Optimization, and Beyond*. MIT Press, Cambridge, MA, USA, 2001.

